# A Method for the Associative Storage
# of Analog Vectors

Amir Atiya (*) and Yaser Abu-Mostafa (**)
(*) Department of Electrical Engineering
(**) Departments of Electrical Engineering and Computer Science
California Institute Technology
Pasadena, Ca 91125

## ABSTRACT

A method for storing analog vectors in Hopfield's continuous feed-back model is proposed. By analog vectors we mean vectors whose components are real-valued. The vectors to be stored are set as equilibria of the network. The network model consists of one layer of visible neurons and one layer of hidden neurons. We propose a learning algorithm, which results in adjusting the positions of the equilibria, as well as guaranteeing their stability. Simulation results confirm the effectiveness of the method.

## 1 INTRODUCTION

The associative storage of binary vectors using discrete feedback neural nets has been demonstrated by Hopfield (1982). This has attracted a lot of attention, and a number of alternative techniques using also the discrete feedback model have appeared. However, the problem of the distributed associative storage of analog vectors has received little attention in literature. By analog vectors we mean vectors whose components are real-valued. This problem is important because in a variety of applications of associative memories like pattern recognition and vector quantization the patterns are originally in analog form and therefore one can save having the costly quantization step and therefore also save increasing the dimension of the vectors. In dealing with analog vectors, we consider feedback networks of the continuous-time graded-output variety, e.g. Hopfield's model (1984):

$$\frac{d\mathbf{u}}{dt} = -\mathbf{u} + \mathbf{W}\mathbf{f}(\mathbf{u}) + \mathbf{a}, \qquad \mathbf{x} = \mathbf{f}(\mathbf{u}), \qquad (1)$$

where $\mathbf{u} = (u_1, ..., u_N)^T$ is the vector of neuron potentials, $\mathbf{x} = (x_1, ..., x_N)^T$ is the vector of firing rates, $\mathbf{W}$ is the weight matrix, $\mathbf{a}$ is the threshold vector, and $\mathbf{f}(\mathbf{u})$ means the vector $(f(u_1), ..., f(u_N))^T$, where $f$ is a sigmoid-shaped function.

The vectors to be stored are set as equilibria of the network. Given a noisy version of any of the stored vectors as the initial state of the network, the network state has

to reach eventually the equilibrium state corresponding to the correct vector. An important requirement is that these equilibria be asymtotically stable, otherwise the attraction to the equilibria will not be guaranteed. Indeed, without enforcing this requirement, our numerical simulations show mostly unstable equilibria.

## 2 THE MODEL

It can be shown that there are strong limitations on the set of memory vectors which can be stored using Hopfield's continuous model (Atiya and Abu-Mostafa 1990). To relieve these limitations, we use an architecture consisting of both visible and hidden units. The outputs of the visible units correspond to the components of the stored vector. Our proposed architecture will be close to the continuous version of the BAM (Kosko 1988). The model consists of one layer of visible units and another layer of hidden units (see Figure 1). The output of each layer is fed as an input to the other layer. No connections exist within each of the layers. Let $\mathbf{y}$ and $\mathbf{x}$ be the output vectors of the hidden layer and the visible layer respectively. Then, in our model,

$$\frac{d\mathbf{u}}{dt} = -\mathbf{u} + \mathbf{W}\mathbf{f}(\mathbf{z}) + \mathbf{a} \equiv \mathbf{e}, \qquad \mathbf{y} = \mathbf{f}(\mathbf{u}) \qquad (2a)$$

$$\frac{d\mathbf{z}}{dt} = -\mathbf{z} + \mathbf{V}\mathbf{f}(\mathbf{u}) + \mathbf{b} \equiv \mathbf{h}, \qquad \mathbf{x} = \mathbf{f}(\mathbf{z}) \qquad (2b)$$

where $\mathbf{W} = [w_{ij}]$ and $\mathbf{V} = [v_{ij}]$ are the weight matrices, $\mathbf{a}$ and $\mathbf{b}$ are the threshold vectors, and $f$ is a sigmoid function (monotonically increasing) in the range from -1 to 1, for example

$$f(u) = \tanh(u).$$

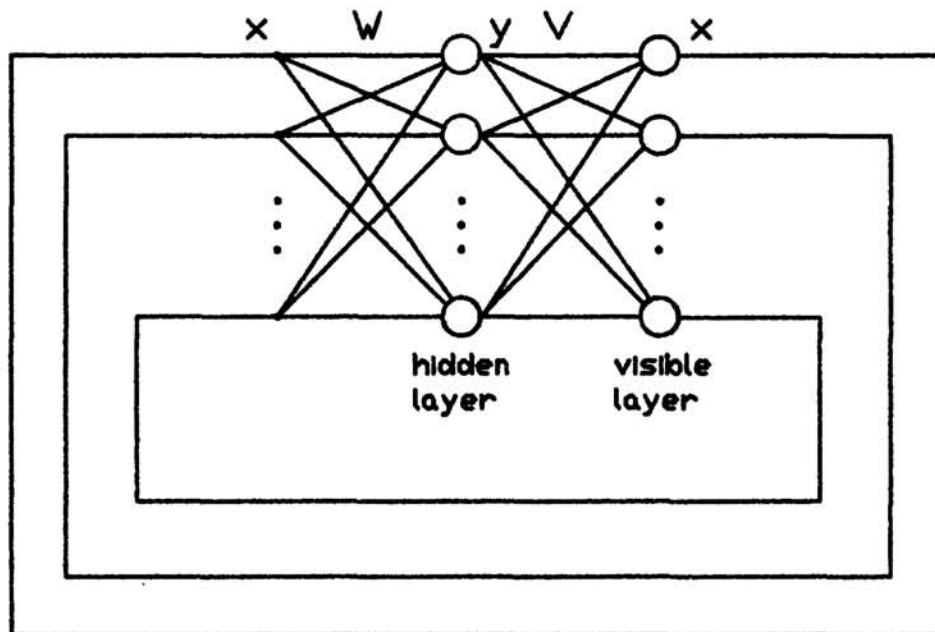

**Figure 1:** The model

As we mentioned before, for a basin of attraction to exist around a given memory vector, the corresponding equilibrium has to be asymtotically stable. For the proposed architecture a condition for stability is given by the following theorem.

**Theorem:** An equilibrium point $(\mathbf{u}^*, \mathbf{z}^*)$ satisfying

$$f'^{1/2}(u_i^*)\sum_j |w_{ij}| f'^{1/2}(z_j^*) < 1 \qquad (3a)$$

$$f'^{1/2}(z_i^*)\sum_j |v_{ij}| f'^{1/2}(u_j^*) < 1 \qquad (3b)$$

for all $i$ is asymptotically stable.

**Proof:** We linearize (2a), (2b) around the equilibrium. We get

$$\frac{d\mathbf{q}}{du} = \mathbf{J}\mathbf{q},$$

where

$$q_i = \begin{cases} u_i - u_i^*, & \text{if } i = 1, ..., N_1 \\ z_{i-N_1} - z_{i-N_1}^*, & \text{if } i = N_1 + 1, ..., N_1 + N_2, \end{cases}$$

$N_1$ and $N_2$ are the number of units in the hidden layer and the visible layer respectively, and $\mathbf{J}$ is the Jacobian matrix, given by

$$\mathbf{J} = \begin{pmatrix} \frac{\partial e_1}{\partial u_1} & \cdots & \frac{\partial e_1}{\partial u_{N_1}} & \frac{\partial e_1}{\partial z_1} & \cdots & \frac{\partial e_1}{\partial z_{N_2}} \\ \vdots & & \vdots & \vdots & & \vdots \\ \frac{\partial e_{N_1}}{\partial u_1} & \cdots & \frac{\partial e_{N_1}}{\partial u_{N_1}} & \frac{\partial e_{N_1}}{\partial z_1} & \cdots & \frac{\partial e_{N_1}}{\partial z_{N_2}} \\ \frac{\partial h_1}{\partial u_1} & \cdots & \frac{\partial h_1}{\partial u_{N_1}} & \frac{\partial h_1}{\partial z_1} & \cdots & \frac{\partial h_1}{\partial z_{N_2}} \\ \vdots & & \vdots & \vdots & & \vdots \\ \frac{\partial h_{N_2}}{\partial u_1} & \cdots & \frac{\partial h_{N_2}}{\partial u_{N_1}} & \frac{\partial h_{N_2}}{\partial z_1} & \cdots & \frac{\partial h_{N_2}}{\partial z_{N_2}} \end{pmatrix},$$

the partial derivatives evaluated at the equilibrium point. Let $\mathbf{\Lambda}_1$ and $\mathbf{\Lambda}_2$ be respectively the $N_1 \times N_1$ and $N_2 \times N_2$ diagonal matrices with the $i^{\text{th}}$ diagonal element being respectively $f'(u_i^*)$ and $f'(z_i^*)$. Furthermore, let

$$\mathbf{\Lambda} = \begin{pmatrix} \mathbf{\Lambda}_1 & \mathbf{0} \\ \mathbf{0} & \mathbf{\Lambda}_2 \end{pmatrix}.$$

The Jacobian is evaluated as

$$\mathbf{J} = \begin{pmatrix} -\mathbf{I}_{N_1} & \mathbf{W}\mathbf{\Lambda}_2 \\ \mathbf{V}\mathbf{\Lambda}_1 & -\mathbf{I}_{N_2} \end{pmatrix}$$

where $\mathbf{I}_L$ means the $L \times L$ identity matrix. Let

$$\mathbf{A} = \begin{pmatrix} -\mathbf{\Lambda}_1^{-1} & \mathbf{W} \\ \mathbf{V} & -\mathbf{\Lambda}_2^{-1} \end{pmatrix}.$$

Then,

$$\mathbf{J} = \mathbf{A}\boldsymbol{\Lambda}.$$

Eigenvalues of $\mathbf{A}\boldsymbol{\Lambda}$ are identical to the eigenvalues of $\boldsymbol{\Lambda}^{1/2}\mathbf{A}\boldsymbol{\Lambda}^{1/2}$ because if $\lambda$ is an eigenvalue of $\mathbf{A}\boldsymbol{\Lambda}$ corresponding to eigenvector $\mathbf{v}$, then

$$\mathbf{A}\boldsymbol{\Lambda}\mathbf{v} = \lambda\mathbf{v},$$

and hence

$$(\boldsymbol{\Lambda}^{1/2}\mathbf{A}\boldsymbol{\Lambda}^{1/2})(\boldsymbol{\Lambda}^{1/2}\mathbf{v}) = \lambda(\boldsymbol{\Lambda}^{1/2}\mathbf{v}).$$

Now, we have

$$\boldsymbol{\Lambda}^{1/2}\mathbf{A}\boldsymbol{\Lambda}^{1/2} = \begin{pmatrix} -\mathbf{I}_{N_1} & \boldsymbol{\Lambda}_1^{1/2}\mathbf{W}\boldsymbol{\Lambda}_2^{1/2} \\ \boldsymbol{\Lambda}_2^{1/2}\mathbf{V}\boldsymbol{\Lambda}_1^{1/2} & -\mathbf{I}_{N_2} \end{pmatrix}.$$

By Gershgorin's Theorem (Franklin 1968), an eigenvalue of $\mathbf{J}$ has to satisfy at least one of the inequalities:

$$|\lambda + 1| \leq f'^{1/2}(u_i^*)\sum_j |w_{ij}| f'^{1/2}(z_j^*) \qquad i = 1, ..., N_1$$

$$|\lambda + 1| \leq f'^{1/2}(z_i^*)\sum_j |v_{ij}| f'^{1/2}(u_j^*) \qquad i = 1, ..., N_2.$$

It follows that under conditions (3a), (3b) that the eigenvalues of $\mathbf{J}$ will have negative real parts, and hence the equilibrium of the original system (2a), (2b) will be asymptotically stable.

Thus, if the hidden unit values are driven far enough into the saturation region (i.e. with values close to 1 or -1), then the corresponding equilibrium will be stable because then, $f'(u_i^*)$ will be very small, causing Inequalities (3) to be satisfied. Although there is nothing to rule out the existence of spurious equilibria and limit cycles, if they occur then they would be far away from the memory vectors because each memory vector has a basin of attraction around it. In our simulations we have never encountered limit cycles.

## 3 TRAINING ALGORITHM

Let $\mathbf{x}^m, m = 1, ..., M$ be the vectors to be stored. Each $\mathbf{x}^m$ should correspond to the visible layer component of one of the asymptotically stable equilibria. We design the network such that the hidden layer component of the equilibrium corresponding to $\mathbf{x}^m$ is far into the saturation region. The target hidden layer component $\mathbf{y}^m$ can be taken as a vector of 1's and -1's, chosen arbitrarily for example by generating the components randomly. Then, the weights have to satisfy

$$y_j^m = f(\sum_l w_{jl} x_l^m + a_j),$$

$$x_i^m = f\left[\sum_j v_{ij} f(\sum_l w_{jl} x_l^m + a_j) + b_i\right].$$

Training is performed in two steps. In the first step we train the weights of the hidden layer. We use steepest descent on the error function

$$E_1 = \sum_{m,j} \|y_j^m - f(\sum_l w_{jl} x_l^m + a_j)\|^2.$$

In the second step we train the weights of the visible layer, using steepest descent on the error function

$$E_2 = \sum_{m,i} \|x_i^m - f[\sum_j v_{ij} f(\sum_l w_{jl} x_l^m + a_j) + b_i]\|^2.$$

We remark that in the first step convergence might be slow since the targets are 1 or -1. A way to have fast convergence is to stop if the outputs are within some constant (say 0.2) from the targets. Then we multiply the weights and the thresholds of the hidden layer by a big positive constant, so as to force the outputs of the hidden layer to be close to 1 or -1.

## 4 IMPLEMENTATION

We consider a network with 10 visible and 10 hidden units. The memory vectors are randomly generated (the components are from -0.8 to 0.8 rather than the full range to have a faster convergence). Five memory vectors are considered. After learning, the memory is tested by giving memory vectors plus noise (100 vectors for a given variance). Figure 2 shows the percentage correct recall in terms of the signal to noise ratio. Although we found that we could store up to 10 vectors, working close to the full capacity is not recommended, as the recall accuracy deteriorates.

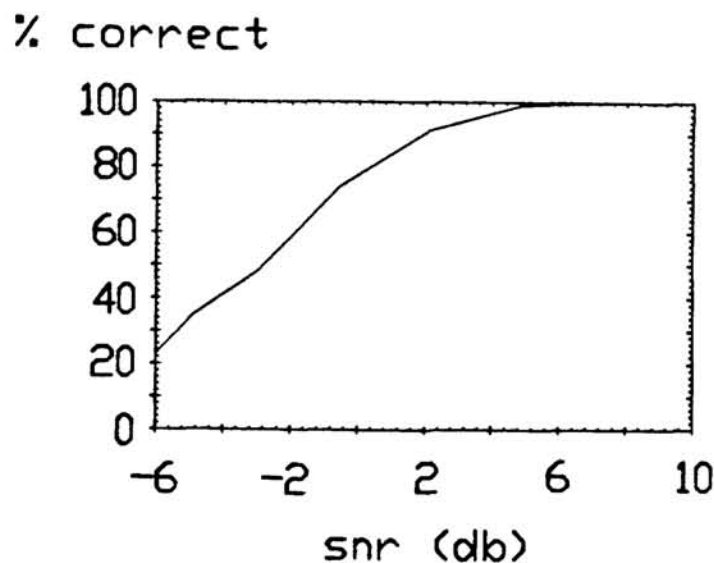

**Figure 2:** Recall accuracy versus signal to noise ratio

## Acknowledgement

This work is supported by the Air Force Office of Scientific Research under grant AFOSR-88-0231.

## References

J. Hopfield (1982), "Neural networks and physical systems with emergent collective computational abilities", *Proc. Nat. Acad. Sci. USA*, vol. 79, pp. 2554-2558.

J. Hopfield (1984), "Neurons with graded response have collective computational properties like those of two state neurons", *Proc. Nat. Acad. Sci. USA*, vol. 81, p. 3088-3092.

A. Atiya and Y. Abu-Mostafa (1990), "An analog feedback associative memory", to be submitted.

B. Kosko (1988), "Bidirectional associative memories", *IEEE Trans. Syst. Man Cybern.*, vol. SMC-18, no. 1, pp. 49-60.

J. Franklin (1968) *Matrix Theory*, Prentice-Hall, Englewood Cliffs, New Jersey.
